# Barycentric Interpolators for Continuous Space & Time Reinforcement Learning

**Rémi Munos & Andrew Moore**
Robotics Institute, Carnegie Mellon University
Pittsburgh, PA 15213, USA.
E-mail:{munos, awm}@cs.cmu.edu

## Abstract

In order to find the optimal control of continuous state-space and time reinforcement learning (RL) problems, we approximate the value function (VF) with a particular class of functions called the *barycentric interpolators*. We establish sufficient conditions under which a RL algorithm converges to the optimal VF, even when we use approximate models of the state dynamics and the reinforcement functions.

## 1 INTRODUCTION

In order to approximate the value function (VF) of a continuous state-space and time reinforcement learning (RL) problem, we define a particular class of functions called the *barycentric interpolator*, that use some interpolation process based on finite sets of points. This class of functions, including continuous or discontinuous piecewise linear and multi-linear functions, provides us with a general method for designing RL algorithms that converge to the optimal value function. Indeed these functions permit us to discretize the HJB equation of the continuous control problem by a *consistent* (and thus convergent) approximation scheme, which is solved by using some model of the state dynamics and the reinforcement functions.

*Section 2* defines the barycentric interpolators. *Section 3* describes the optimal control problem in the deterministic continuous case. *Section 4* states the convergence result for RL algorithms by giving sufficient conditions on the applied model. *Section 5* gives some computational issues for this method, and *Section 6* describes the approximation scheme used here and proves the convergence result.

## 2  DEFINITION OF BARYCENTRIC INTERPOLATORS

Let $\Sigma^\delta = \{\xi_i\}_i$ be a set of points distributed at some resolution $\delta$ (see (4) below) on the state space of dimension $d$.

For any state $x$ inside some simplex $(\xi_1, ..., \xi_n)$, we say that $x$ is the *barycenter* of the $\{\xi_i\}_{i=1..n}$ inside this simplex with positive coefficients $p(x|\xi_i)$ of sum 1, called the *barycentric coordinates*, if $x = \sum_{i=1..n} p(x|\xi_i).\xi_i$.

Let $V^\delta(\xi_i)$ be the value of the function at the points $\xi_i$. $V^\delta$ is a **barycentric interpolator** if for any state $x$ which is the barycenter of the points $\{\xi_i\}_{i=1..n}$ for some simplex $(\xi_1, ..., \xi_n)$, with the barycentric coordinates $p(x|\xi_i)$, we have :

$$V^\delta(x) = \sum_{i=1..n} p(x|\xi_i).V^\delta(\xi_i) \tag{1}$$

Moreover we assume that the simplex $(\xi_1, ..., \xi_n)$ is of diameter $O(\delta)$. Let us describe some simple barycentric interpolators :

- **Piecewise linear functions** defined by some triangulation on the state space (thus defining continuous functions), see figure 1.a, or defined at any $x$ by a linear combination of $(d+1)$ values at any points $(\xi_1, ..., \xi_{d+1}) \ni x$ (such functions may be discontinuous at some boundaries), see figure 1.b.

- **Piecewise multi-linear functions** defined by a multi-linear combination of the $2^d$ values at the vertices of $d$-dimensional rectangles, see figure 1.c. In this case as well, we can build continuous interpolations or allow discontinuities at the boundaries of the rectangles.

An important point is that the convergence result stated in Section 4 does not require the continuity of the function. This permits us to build variable resolution triangulations (see figure 1.b) or grid (figure 1.c) easily.

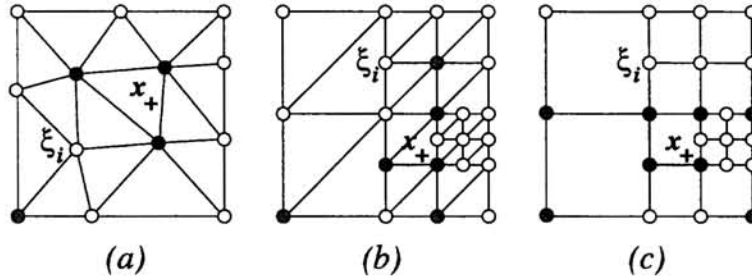

*(a)*          *(b)*          *(c)*

Figure 1: Some examples of barycentric approximators. These are piecewise continuous $(a)$ or discontinuous $(b)$ linear or multi-linear $(c)$ interpolators.

**Remark 1** *In the general case, for a given $x$, the choice of a simplex $(\xi_1, ..., \xi_n) \ni x$ is not unique (see the two sets of grey and black points in figure 1.b and 1.c), and once the simplex $(\xi_1, ..., \xi_n) \ni x$ is defined, if $n > d + 1$ (for example in figure 1.c), then the choice of the barycentric coordinates $p(x|\xi_i)$ is also not unique.*

**Remark 2** *Depending on the interpolation method we use, the time needed for computing the values will vary. Following [Dav96], the continuous multi-linear interpolation must process $2^d$ values, whereas the linear continuous interpolation inside a simplex processes $(d + 1)$ values in $O(d \log d)$ time.*

In comparison to [Gor95], the functions used here are *averagers* that satisfy the barycentric interpolation property (1). This additional geometric constraint permits us to prove the consistency (see (15) below) of the approximation scheme and thus the convergence to the optimal value in the continuous time case.

## 3 THE OPTIMAL CONTROL PROBLEM

Let us describe the optimal control problem in the *deterministic* and *discounted* case for *continuous state-space* and *time* variables and define the value function that we intend to approximate. We consider a dynamical system whose *state dynamics* depends on the *current state* $x(t) \in \bar{O}$ (the *state-space*, with $O$ an open subset of $\mathbb{R}^d$) and *control* $u(t) \in U$ (compact subset) by a differential equation :

$$\frac{dx}{dt} = f(x(t), u(t)) \tag{2}$$

From equation (2), the choice of an initial state $x$ and a control function $u(t)$ leads to a unique trajectories $x(t)$ (see figure 2). Let $\tau$ be the *exit time* from $\overline{O}$ (with the convention that if $x(t)$ always stays in $\overline{O}$, then $\tau = \infty$). Then, we define the *functional J* as the discounted cumulative reinforcement :

$$J(x; u(.)) = \int_0^\tau \gamma^t r(x(t), u(t)) dt + \gamma^\tau R(x(\tau))$$

where $r(x, u)$ is the *running reinforcement* and $R(x)$ the *boundary reinforcement*. $\gamma$ is the *discount factor* ($0 \leq \gamma < 1$). We assume that $f$, $r$ and $R$ are bounded and Lipschitzian, and that the boundary $\partial O$ is $C^2$.

RL uses the method of Dynamic Programming (DP) that introduces the *value function* (VF) : the maximal value of $J$ as a function of initial state $x$ :

$$V(x) = \sup_{u(.)} J(x; u(.)).$$

From the DP principle, we deduce that $V$ satisfies a first-order differential equation, called the Hamilton-Jacobi-Bellman (HJB) equation (see [FS93] for a survey) :

**Theorem 1** *If $V$ is differentiable at $x \in O$, let $DV(x)$ be the gradient of $V$ at $x$, then the following HJB equation holds at $x$.*

$$H(V, DV, x) \stackrel{\text{def}}{=} V(x) \ln \gamma + \sup_{u \in U}[DV(x).f(x, u) + r(x, u)] = 0 \tag{3}$$

The challenge of RL is to get a good approximation of the VF, because from $V$ we can deduce the optimal control : for state $x$, the control $u^*(x)$ that realizes the supremum in the HJB equation provides an optimal (feed-back) control law.

The following hypothesis is a sufficient condition for $V$ to be continuous within $O$ (see [Bar94]) and is required for proving the convergence result of the next section.

**Hyp 1:** For $x \in \partial O$, let $\overrightarrow{n}(x)$ be the outward normal of $O$ at $x$, we assume that :
  *-If $\exists u \in U$, s.t. $f(x, u).\overrightarrow{n}(x) \leq 0$ then $\exists v \in U$, s.t. $f(x, v)\overrightarrow{n}(x) < 0$.*
  *-If $\exists u \in U$, s.t. $f(x, u).\overrightarrow{n}(x) \geq 0$ then $\exists v \in U$, s.t. $f(x, v)\overrightarrow{n}(x) > 0$.*

which means that at the states (if there exist any) where some trajectory is tangent to the boundary, there exists, for some control, a trajectory strictly coming inside and one strictly leaving the state space.

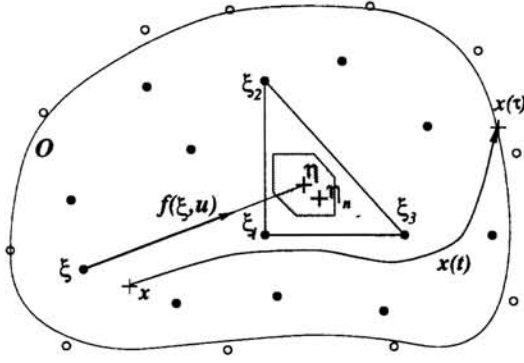

Figure 2: The state space and the set of points $\Sigma^\delta$ (the black dots belong to the interior and the white ones to the boundary). The value at some point $\xi$ is updated, at step $n$, by the discounted value at point $\eta_n \in (\xi_1, \xi_2, \xi_3)$. The main requirement for convergence is that the points $\eta_n$ approximate $\eta$ in the sense : $p(\eta_n|\xi_i) = p(\eta|\xi_i) + O(\delta)$ (i.e. the $\eta_n$ belong to the grey area).

## 4   THE CONVERGENCE RESULT

Let us introduce the set of points $\Sigma^\delta = \{\xi_i\}_i$, composed of the *interior* $(\Sigma^\delta \cap O)$ and the *boundary* $(\partial\Sigma^\delta = \Sigma^\delta \setminus O)$, such that its convex hull covers the state space $\overline{O}$, and performing a discretization at some resolution $\delta$ :

$$\forall x \in O, \inf_{\xi_i \in \Sigma^\delta \cap O} ||x - \xi_i|| \leq \delta \quad \text{and} \quad \forall x \in \partial O \inf_{\xi_j \in \partial\Sigma^\delta} ||x - \xi_j|| \leq \delta \tag{4}$$

Moreover, we approximate the control space $U$ by some finite control spaces $U^\delta \subset U$ such that for $\delta \leq \delta'$, $U^{\delta'} \subset U^\delta$ and $\lim_{\delta \to 0} U^\delta = U$.

We would like to update the value of any :

- *interior point* $\xi \in \Sigma^\delta \cap O$ with the discounted values at state $\eta_n(\xi, u)$ (figure 2) :

$$V^\delta_{n+1}(\xi) \leftarrow \sup_{u \in U^\delta} \left[ \gamma^{\tau_n(\xi,u)} V^\delta_n(\eta_n(\xi, u)) + \tau_n(\xi, u).r_n(\xi, u) \right] \tag{5}$$

for some state $\eta_n(\xi, u)$, some time delay $\tau_n(\xi, u)$ and some reinforcement $r_n(\xi, u)$.

- *boundary point* $\xi \in \partial\Sigma^\delta$ with some terminal reinforcement $R_n(\xi)$ :

$$V^\delta_{n+1}(\xi) \leftarrow R_n(\xi) \tag{6}$$

The following theorem states that the values $V^\delta_n$ computed by a RL algorithm using the model (because of some a priori partial uncertainty of the state dynamics and the reinforcement functions) $\eta_n(\xi, u)$, $\tau_n(\xi, u)$, $r_n(\xi, u)$ and $R_n(\xi)$ converge to the optimal value function as the number of iterations $n \to \infty$ and the resolution $\delta \to 0$.

Let us define the state $\eta(\xi, u)$ (see figure 2) :

$$\eta(\xi, u) = \xi + \tau(\xi, u).f(\xi, u) \tag{7}$$

for some time delay $\tau(\xi, u)$ (with $k_1\delta \leq \tau(\xi, u) \leq k_2\delta$ for some constants $k_1 > 0$ and $k_2 > 0$), and let $p(\eta|\xi_i)$ (resp. $p(\eta_n|\xi_i)$) be the barycentric coordinate of $\eta$ inside a simplex containing it (resp. $\eta_n$ inside the same simplex). We will write $\eta$, $\eta_n$, $\tau$, $r$, ..., instead of $\eta(\xi, u)$, $\eta_n(\xi, u)$, $\tau(\xi, u)$, $r(\xi, u)$, ... when no confusion is possible.

**Theorem 2** *Assume that the hypotheses of the previous sections hold, and that for any resolution $\delta$, we use barycentric interpolators $V^\delta$ defined on state spaces $\Sigma^\delta$ (satisfying (4)) such that all points of $\Sigma^\delta \cap O$ are regularly updated with rule (5) and all points of $\partial\Sigma^\delta$ are updated with rule (6) at least once. Suppose that $\eta_n$, $\tau_n$, $r_n$ and $R_n$ approximate $\eta$, $\tau$, $r$ and $R$ in the sense :*

$$\forall \xi_i, \ p(\eta_n|\xi_i) = p(\eta|\xi_i) + O(\delta) \tag{8}$$
$$\tau_n = \tau + O(\delta^2) \tag{9}$$
$$r_n = r + O(\delta) \tag{10}$$
$$R_n = R + O(\delta) \tag{11}$$

*then we have* $\lim_{\substack{n \to \infty \\ \delta \to 0}} V_n^\delta = V$ *uniformly on any compact* $\Omega \subset O$ *(i.e.* $\forall \varepsilon > 0, \forall \Omega$
*compact* $\subset O, \exists \Delta, \exists N$, *such that* $\forall \delta \leq \Delta, \forall n \geq N, \sup_{\Sigma^\delta \cap \Omega} |V_n^\delta - V| \leq \varepsilon$).

**Remark 3** *For a given value of* $\delta$, *the rule (5) is not a DP updating rule for some Markov Decision Problem (MDP) since the values* $\eta_n, \tau_n, r_n$ *depend on* $n$. *This point is important in the RL framework since this allows on-line improvement of the model of the state dynamics and the reinforcement functions.*

**Remark 4** *This result extends the previous results of convergence obtained by Finite-Element or Finite-Difference methods (see [Mun97]).*

This theoretical result can be applied by starting from a rough $\Sigma^\delta$ (high $\delta$) and by combining to the iteration process ($n \to \infty$) some learning process of the model ($\eta_n \to \eta$) and a increasing process of the number of points ($\delta \to 0$).

## 5   COMPUTATIONAL ISSUES

From (8) we deduce that the method will also converge if we use an *approximate barycentric interpolator*, defined at any state $x \in (\xi_1, ..., \xi_n)$ by the value of the barycentric interpolator at some state $x' \in (\xi_1, ..., \xi_n)$ such that $p(x'|\xi_i) = p(x|\xi_i) + O(\delta)$ (see figure 3). The fact that we need not be completely accurate can be

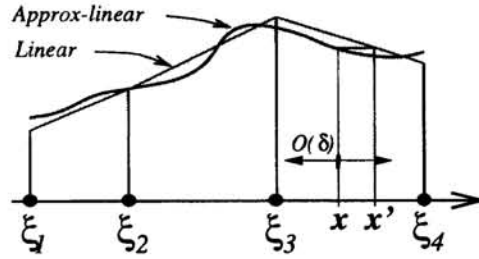

Figure 3: The linear function and the approximation error around it (the grey area). The value of the approximate linear function plotted here at some state $x$ is equal to the value of the linear one at $x'$. Any such approximate barycenter interpolator can be used in (5).

used to our advantage. First, the computation of barycentric coordinates can use very fast approximate matrix methods. Second, the model we use to integrate the dynamics need not be perfect. We can make an $O(\delta^2)$ error, which is useful if we are learning a model from data: we need simply arrange to not gather more data than is necessary for the current $\delta$. For example, if we use nearest neighbor for our dynamics learning, we need to ensure enough data so that every observation is $O(\delta^2)$ from its nearest neighbor. If we use local regression, then a mere $O(\delta)$ density is all that is required [Omo87, AMS97].

## 6   PROOF OF THE CONVERGENCE RESULT

### 6.1   Description of the approximation scheme

We use a convergent scheme derived from Kushner (see [Kus90]) in order to approximate the continuous control problem by a finite MDP. The HJB equation is discretized, at some resolution $\delta$, into the following DP equation : for $\xi \in \Sigma^\delta \cap O$,

$$V^\delta(\xi) = F^\delta \left[ V^\delta(.) \right](\xi) \overset{\text{def}}{=} \sup_{u \in U^\delta} \left\{ \gamma^\tau \sum_{\xi_i} p(\eta|\xi_i).V^\delta(\xi_i) + \tau.r \right\} \quad (12)$$

and for $\xi \in \partial \Sigma^\delta$, $V^\delta(\xi) = R(\xi)$. This is a fixed-point equation and we can prove that, thanks to the discount factor $\gamma$, it satisfies the **"strong" contraction** property :

$$\sup_{\Sigma^\delta} \left| V_{n+1}^\delta - V^\delta \right| \leq \lambda . \sup_{\Sigma^\delta} \left| V_n^\delta - V^\delta \right| \text{ for some } \lambda < 1 \quad (13)$$

from which we deduce that there exists exactly one solution $V^\delta$ to the DP equation, which can be computed by some value iteration process : for any initial $V_0^\delta$, we iterate $V_{n+1}^\delta \leftarrow F^\delta \left[ V_n^\delta \right]$. Thus for any resolution $\delta$, the values $V_n^\delta \to V^\delta$ as $n \to \infty$.

Moreover, as $V^\delta$ is a barycentric interpolator and from the definition (7) of $\eta$,

$$F^\delta \left[ V^\delta (.) \right] (\xi) = \sup_{u \in U^\delta} \left\{ \gamma^\tau V^\delta (\xi + \tau . f(\xi, u)) + \tau . r \right\} \qquad (14)$$

from which we deduce that the scheme $F^\delta$ is *consistent* : in a formal sense,

$$\limsup_{\delta \to 0} \tfrac{1}{\delta} |F^\delta [W](x) - W(x)| \sim H(W, DW, x) \qquad (15)$$

and obtain, from the general convergence theorem of [BS91] (and a result of strong unicity obtained from hyp.1), the convergence of the scheme : $V^\delta \to V$ as $\delta \to 0$.

## 6.2   Use of the "weak contraction" result of convergence

Since in the RL approach used here, we only have an approximation $\eta_n$, $\tau_n$, ... of the true values $\eta$, $\tau$, ..., the strong contraction property (13) does not hold any more. However, in previous work ([Mun98]), we have proven the convergence for some weakened conditions, recalled here :

If the values $V_n^\delta$ updated by some algorithm satisfy the **"weak"** contraction property with respect to a solution $V^\delta$ of a convergent approximation scheme (such as the previous one (12)) :

$$\sup_{\Sigma^\delta \cap O} \left| V_{n+1}^\delta - V^\delta \right| \quad \le \quad (1 - k.\delta). \sup_{\Sigma^\delta} \left| V_n^\delta - V^\delta \right| + o(\delta) \qquad (16)$$

$$\sup_{\partial \Sigma^\delta} \left| V_{n+1}^\delta - V^\delta \right| \quad = \quad O(\delta) \qquad (17)$$

for some positive constant $k$, (with the notation $f(\delta) \le o(\delta)$ iff $\exists g(\delta) = o(\delta)$ with $f(\delta) \le g(\delta)$) then we have $\lim_{\substack{n \to \infty \\ \delta \to 0}} V_n^\delta = V$ uniformly on any compact $\Omega \subset O$ (i.e. $\forall \varepsilon > 0$, $\forall \Omega$ compact $\subset O$, $\exists \Delta$ and $N$ such that $\forall \delta \le \Delta, \forall n \ge N$, $\sup_{\Sigma^\delta \cap \Omega} \left| V_n^\delta - V \right| \le \varepsilon$).

## 6.3   Proof of theorem 2

We are going to use the approximations (8), (9), (10) and (11) to deduce that the weak contraction property holds, and then use the result of the previous section to prove theorem 2.

The proof of (17) is immediate since, from (6) and (11) we have : $\forall \xi \in \partial \Sigma^\delta$,

$$\left| V_{n+1}^\delta (\xi) - V^\delta (\xi) \right| = |R_n(\xi) - R(\xi)| = O(\delta)$$

Now we need to prove (16). Let us estimate the error $E_n(\xi) = V^\delta(\xi) - V_n^\delta(\xi)$ between the value $V^\delta$ of the DP equation (12) and the values $V_n^\delta$ computed by rule (5) after one iteration :

$$E_{n+1}(\xi) = \sup_{u \in U^\delta} \left\{ \sum_{\xi_i} \left[ \gamma^\tau p(\eta | \xi_i) . V^\delta(\xi_i) - \gamma^{\tau_n} p(\eta_n | \xi_i) . V_n^\delta(\xi_i) \right] + \tau . r - \tau_n . r_n \right\}$$

$$E_{n+1}(\xi) = \sup_{u \in U^\delta} \left\{ \gamma^\tau \sum_{\xi_i} \left[ p(\eta | \xi_i) - p(\eta_n | \xi_i) \right] V^\delta(\xi_i) + \left[ \gamma^\tau - \gamma^{\tau_n} \right] \sum_{\xi_i} p(\eta_n | \xi_i) . V^\delta(\xi_i) \right.$$
$$\left. + \gamma^{\tau_n} \sum_{\xi_i} p(\eta_n | \xi_i) . \left[ V^\delta(\xi_i) - V_n^\delta(\xi_i) \right] + \tau_n \left[ r - r_n \right] + \left[ \tau - \tau_n \right] r \right\}$$

By using (9) (from which we deduce : $\gamma^\tau = \gamma^{\tau_n} + O(\delta^2)$) and (10), we deduce :

$$|E_{n+1}(\xi)| \quad \le \quad \sup_{u \in U^\delta} \left\{ \gamma^\tau . \left| \sum_{\xi_i} \left[ p(\eta | \xi_i) - p(\eta_n | \xi_i) \right] V^\delta(\xi_i) \right| \qquad (18) \right.$$

$$\left. + \gamma^{\tau_n} \sum_{\xi_i} p(\eta_n | \xi_i) . \left| V^\delta(\xi_i) - V_n^\delta(\xi_i) \right| \right\} + O(\delta^2).$$

From the basic properties of the coefficients $p(\eta|\xi_i)$ and $p(\eta_n|\xi_i)$ we have :

$$\sum_{\xi_i} [p(\eta|\xi_i) - p(\eta_n|\xi_i)] V^\delta(\xi_i) = \sum_{\xi_i} [p(\eta|\xi_i) - p(\eta_n|\xi_i)] [V^\delta(\xi_i) - V^\delta(\xi)] \quad (19)$$

Moreover, $|V^\delta(\xi_i) - V^\delta(\xi)| \leq |V^\delta(\xi_i) - V(\xi_i)| + |V(\xi_i) - V(\xi)| + |V(\xi) - V^\delta(\xi)|$.
From the convergence of the scheme $V^\delta$, we have $\sup_{\Sigma^\delta \cap \Omega} |V^\delta - V| \overset{\delta \downarrow 0}{\to} 0$ for any compact $\Omega \subset O$ and from the continuity of $V$ and the fact that the support of the simplex $\{\xi\} \ni \eta$ is $O(\delta)$, we have $\sup_{\Sigma^\delta \cap \Omega} |V(\xi_i) - V(\xi)| \overset{\delta \downarrow 0}{\to} 0$ and deduce that :
$\sup_{\Sigma^\delta \cap \Omega} |V^\delta(\xi_i) - V^\delta(\xi)| \overset{\delta \downarrow 0}{\to} 0$. Thus, from (19) and (8), we obtain :

$$\left| \sum_{\xi_i} [p(\eta|\xi) - p(\eta_n|\xi)] V^\delta(\xi_i) \right| = o(\delta) \quad (20)$$

**The "weak" contraction property (16) holds :** from the property of the exponential function $\gamma^{\tau_n} \leq 1 - \frac{\tau_n}{2} \ln \frac{1}{\gamma}$ for small values of $\tau_n$, from (9) and that $\tau \geq k_1 \delta$, we deduce that $\gamma^{\tau_n} \leq 1 - \frac{k_1 \delta}{2} \ln \frac{1}{\gamma} + O(\delta^2)$, and from (18) and (20) we deduce that :

$$\left| V_{n+1}^\delta(\xi) - V^\delta(\xi) \right| \leq (1 - k.\delta) \sup_{\Sigma^\delta} \left| V_{n+1}^\delta(\xi) - V^\delta(\xi) \right| + o(\delta)$$

with $k = \frac{k_1 \delta}{2} \ln \frac{1}{\gamma}$, and the property (16) holds. Thus the "weak contraction" result of convergence (described in section 6.2) applies and convergence occurs.

## FUTURE WORK

This work proves the convergence to the optimal value as the resolution tends to the limit, but does not provide us with the rate of convergence. Our future work will focus on defining upper bounds of the approximation error, especially for variable resolution discretizations, and we will also consider the stochastic case.

## ACKNOWLEDGMENTS

This research was sponsored by DASSAULT-AVIATION and CMU.

## References

[AMS97]  C. G. Atkeson, A. W. Moore, and S. A. Schaal. Locally Weighted Learning. *AI Review*, 11:11–73, April 1997.

[Bar94]  Guy Barles. *Solutions de viscosité des équations de Hamilton-Jacobi*, volume 17 of *Mathématiques et Applications*. Springer-Verlag, 1994.

[BS91]  Guy Barles and P.E. Souganidis. Convergence of approximation schemes for fully nonlinear second order equations. *Asymptotic Analysis*, 4:271–283, 1991.

[Dav96]  Scott Davies. Multidimensional triangulation and interpolation for reinforcement learning. *Advances in Neural Information Processing Systems*, 8, 1996.

[FS93]  Wendell H. Fleming and H. Mete Soner. *Controlled Markov Processes and Viscosity Solutions*. Applications of Mathematics. Springer-Verlag, 1993.

[Gor95]  G. Gordon. Stable function approximation in dynamic programming. *International Conference on Machine Learning*, 1995.

[Kus90]  Harold J. Kushner. Numerical methods for stochastic control problems in continuous time. *SIAM J. Control and Optimization*, 28:999–1048, 1990.

[Mun97]  Rémi Munos. A convergent reinforcement learning algorithm in the continuous case based on a finite difference method. *International Joint Conference on Artificial Intelligence*, 1997.

[Mun98]  Rémi Munos. A general convergence theorem for reinforcement learning in the continuous case. *European Conference on Machine Learning*, 1998.

[Omo87]  S. M. Omohundro. Efficient Algorithms with Neural Network Behaviour. *Journal of Complex Systems*, 1(2):273–347, 1987.